# Understanding the Intrinsic Memorability of Images

**Phillip Isola**
MIT
phillipi@mit.edu

**Devi Parikh**
TTI-Chicago
dparikh@ttic.edu

**Antonio Torralba**
MIT
torralba@mit.edu

**Aude Oliva**
MIT
oliva@mit.edu

## Abstract

Artists, advertisers, and photographers are routinely presented with the task of creating an image that a viewer will remember. While it may seem like image memorability is purely subjective, recent work shows that it is not an inexplicable phenomenon: variation in memorability of images is consistent across subjects, suggesting that some images are *intrinsically* more memorable than others, independent of a subjects' contexts and biases. In this paper, we used the publicly available memorability dataset of Isola *et al.* [13], and augmented the object and scene annotations with interpretable spatial, content, and aesthetic image properties. We used a feature-selection scheme with desirable explaining-away properties to determine a compact set of attributes that characterizes the memorability of any individual image. We find that images of enclosed spaces containing people with visible faces are memorable, while images of vistas and peaceful scenes are not. Contrary to popular belief, unusual or aesthetically pleasing scenes do not tend to be highly memorable. This work represents one of the first attempts at understanding intrinsic image memorability, and opens a new domain of investigation at the interface between human cognition and computer vision.

## 1 Introduction

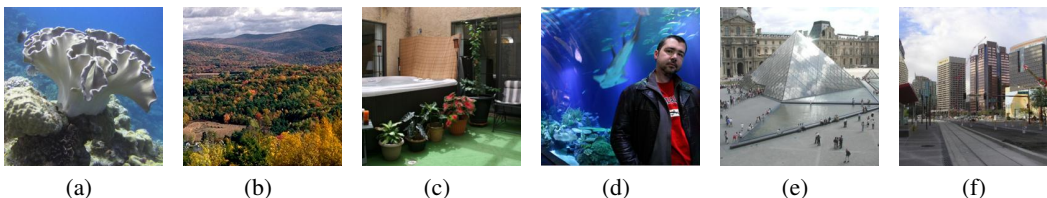

| (a) | (b) | (c) | (d) | (e) | (f) |

Figure 1: Which of these images are the most memorable? See footnote 1 for the answer key.

When glancing at a magazine or browsing the Internet we are continuously exposed to photographs and images. Despite this overflow of visual information, humans are extremely good at remembering thousands of pictures and a surprising amount of their visual details [1, 15, 16, 25, 30]. But, while some images stick in our minds, others are ignored or quickly forgotten. Artists, advertisers, and photographers are routinely challenged by the question "what makes an image memorable?" and are then presented with the task of creating an image that will be remembered by the viewer.

While psychologists have studied human capacity to remember visual stimuli [1,15,16,25,30], little work has systematically studied the differences in stimuli that make them more or less memorable. In a recent paper [13], we quantified the memorability of 2222 photographs as the rate at which subjects detect a repeat presentation of the image a few minutes after its initial presentation. The memorability of these images was found to be consistent across subjects and across a variety of contexts, making some of these images *intrinsically* more memorable than others, independent of the subjects' past experiences or biases. Thus, while image memorability may seem like a quality that is hard to quantify, our recent work suggests that it is not an inexplicable phenomenon.

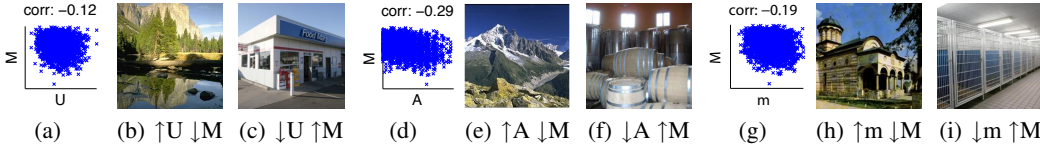

Figure 2: Distribution of memorability M of photographs with respect to unusualness U (left), aesthetics A (middle) and subjects' guess on how memorable an image is m (right). All 2222 images from the memorability dataset were rated along these three aspects by 10 subjects each. Contrary to popular belief, unusual and aesthetically pleasing images are not predominantly the most memorable ones. Also shown are example images that demonstrate this (*e.g.* (f) shows an image that is very aesthetic, but not memorable). Clearly, which images are memorable is not intuitive, as seen by poor estimates from subjects (g).

But then again, subjective intuitions of what make an image memorable may need to be revised. For instance, look at the photographs of Figure 1. Which images do you think are more memorable?[1] We polled various human and computer vision experts to get ideas as to what people think drives memorability. Among the most frequent responses were unusualness (8 out of 16) and aesthetic beauty (7 out of 16). Surprisingly, as shown in Figure 2, we find that these are weakly correlated (and, in fact, negatively correlated) with memorability as measured in [13]. Further, when subjects were asked to rate how memorable they think an image would be, their responses were weakly (negatively) correlated to true memorability (Figure 2)!

While our previous work aimed at predicting memorability [13], here we aim to better *understand* memorability. Any realistic use of the memorability of images requires an understanding of the key factors that underly memorability; be it for cognitive scientists to discover the mechanisms behind memory or for advertisement designers to create more effective visual media.

Thus, the goal of this paper is to identify a collection of human-understandable visual attributes that are highly informative about image memorability. First, we annotate the memorability dataset [13] with interpretable and semantic attributes. Second, we employ a greedy feature selection algorithm with desirable explaining-away properties that allows us to explicitly determine a compact set of characteristics that make an image memorable. Finally, we train automatic detectors that predict these characteristics, which are in turn used to predict memorability.

## 2 Related work

**Visual memory:** People have been shown to have a remarkable ability to remember particular images in long-term memory, be they everyday scenes, objects and events [30], or the shapes of arbitrary forms [25]. As most of us would expect, image memorability depends on the user context and is likely to be subject to some inter-subject variability [12]. However, in our previous work [13], we found that despite this expected variability, there is also a large degree of agreement between users. This suggests that there is something intrinsic to images that make some more memorable than others, and in [13] we developed a computer vision algorithm to predict this intrinsic memorability. While being a useful goal, prediction systems are often uninterpretable, giving us little insight into what makes the image memorable. Hence in this work, we focus on identifying the characteristics of images that make them memorable. A discussion of different models of memory retrieval [3,11,27] and formation [22] are beyond the scope of this paper.

**Attributes for interpretability:** Attributes-based visual recognition has received a lot of attention in computer vision literature in recent years. Attributes can be thought of as mid-level interpretable features such as "furry" and "spacious". Attributes are attractive because they allow for transfer-learning among categories that share attributes [18]. Attributes also allow for descriptions of previously unseen images [8]. In this work, we exploit attributes to understand which properties of an image make it memorable.

**Predicting image properties:** While image memorability is vastly unexplored, many other photographic properties have been studied in the literature, such as photo quality [21], saliency [14], attractiveness [20], composition [10, 24], color harmony [5], and object importance [29]. Most related to our work is the recent work of Dhar *et al.* [7], who use attributes to predict the aesthetic quality of an image. Towards the goal of improved prediction, they use a list of attributes known to influence the aesthetic quality of an image. In our work, since it is not known what makes an image

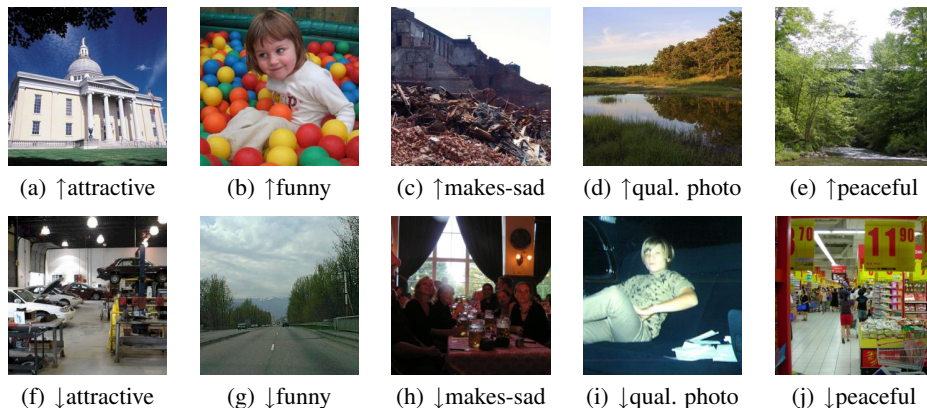

(a) ↑attractive  (b) ↑funny  (c) ↑makes-sad  (d) ↑qual. photo  (e) ↑peaceful

(f) ↓attractive  (g) ↓funny  (h) ↓makes-sad  (i) ↓qual. photo  (j) ↓peaceful

Figure 3: Example images depicting varying values of a subset of attributes annotated by subjects.

memorable, we use an exhaustive list of attributes, and use a feature selection scheme to *identify* which attributes make an image memorable.

## 3  Attribute annotations

We investigate memorability using the memorability dataset from [13]. The dataset consists of 2222 natural images of everyday scenes and events selected from the SUN dataset [32], as well as memorability scores for each image. The memorability scores were obtained via 665 subjects playing a 'memory game' on Amazon's Mechanical Turk. A series of natural images were flashed for 1 second each. Subjects were instructed to press a key whenever they detected a repeat presentation of an image. The memorability score of an image corresponds to the number of subjects that correctly detected a repeat presentation of the image. The rank correlation between two halves of the subjects was found to be 0.75, providing evidence for intrinsic image memorability. Examples images from this dataset can be seen throughout the paper.

The images in the memorability dataset come from ~700 scene categories [32]. They have been labeled via the LabelMe [26] online annotation tool, and contain about ~1300 object categories. While the scene and object categories depicted in an image may very well influence its memorability, there are many other properties of an image that could be at play. To get a handle on these, we constructed an extensive list of image properties or attributes, and had the 2222 images annotated with these properties using Amazon's Mechanical Turk. An organization of the attributes collected is shown in Table 1. Binary attributes are listed with a '?', while multi-valued attributes (on a scale of 1-5) are listed with a ';'. Each image was annotated by 10 subjects for each of the attributes. The average response across the subjects was stored as the value of the attribute for an image. The 'Length of description' attribute was computed as the average number of words subjects used to describe the image (free-form). The spatial layout attributes were based on the work of Oliva and Torralba [23]. Many of the aesthetic attributes are based on the work of Dhar *et al.* [7].

We noticed that images containing people tend to be highly memorable. However even among images containing people, there is a variation in memorability that is consistent across subjects (split half rank correlation = 0.71). In an effort to better understand memorability of images containing people, we collected several attributes that are specific to people. These are listed in Table 2. The annotations of these attributes were collected only on images containing people (and are considered to be absent for images not containing people). This is compactly captured by the 'contains a person' attribute.

Some questions had multiple choice answers (for example, Age can take four values: child, teenager, adult and senior). When applicable, the multiple choices are listed in parentheses in Table 2. Each choice was treated as a separate binary attribute (*e.g.* is child). Some of the people-attributes were referring to the entire image ('whole image') while others were referring to each person in the image ('per-person'). The per-person attributes were aggregated across all subjects and all people in the image. See Figure 3 for example attribute annotations.

Table 1: General attributes

| |
|---|
| **Spatial layout:** Enclosed space vs. Open space; Perspective view vs. Flat view; Empty space vs. Cluttered space; Mirror symmetry vs. No mirror symmetry (cf. [23]) |
| **Aesthetics:** Post-card like? Buy this painting? Hang-on wall? Is aesthetic? Pleasant vs. Unpleasant; Unusual or strange vs. Routine or mundane; Boring vs. Striking colors; High quality (expert photography) vs. Poor quality photo; Attractive vs. Dull photo; Memorable vs. Not memorable; Sky present? Clear vs. Cloudy sky; Blue vs. Sunset sky; Picture of mainly one object vs. Whole scene; Single focus vs. Many foci; Zoomed-in vs. Zoomed-out; Top down view vs. Side view (cf. [7]) |
| **Emotions:** Frightening? Arousing? Funny? Engaging? Peaceful? Exciting? Interesting? Mysterious? Strange? Striking? Makes you happy? Makes you sad? |
| **Dynamics:** Action going on? Something moving in scene? Picture tells a story? About to happen? Lot going on? Dynamic scene? Static scene? Have a lot to say; Length of description |
| **Location:** Famous place? Recognize place? Like to be present in scene? Many people go here? |
| **Contains a person?** |

For further analysis, we utilize the most frequent 106 of the ∼1300 objects present in the images (their presence, count, area in the image, and for a subset of these objects, area occupied in four quadrants of the image), 237 of the ∼700 scene categories, and the 127 attributes listed in Tables 1 and 2. We also append image annotations with a scene hierarchy provided with the SUN dataset [32] that groups similar categories into a meta-category (*e.g.*indoor), as well as an object hierarchy derived from the WordNet [9], that includes meta-categories such as organism and furniture. The scene hierarchy resulted in 19 additional scene meta-categories, while the object hierarchy resulted in 134 additional meta-categories. From here on, we will refer to all these annotations as features. We have a total of 923 features. The goal now is to determine a concise subset of these features that characterizes the memorability of an image. Since all our features are human-interpretable, this allows us to gain an understanding of what makes an image memorable. Figure 4 shows the correlation of different feature types with memorability.

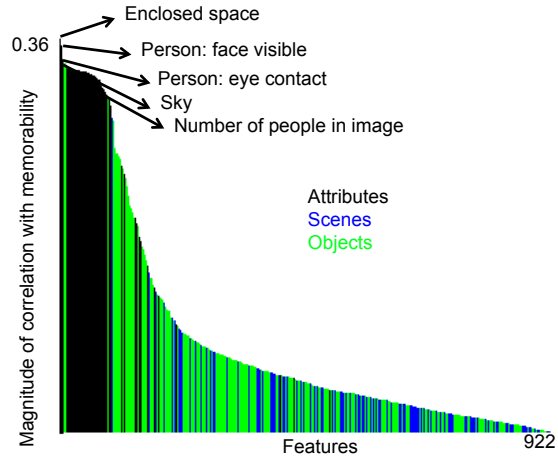

Figure 4: Correlation of attribute, scene, and object annotations with memorability. We see that the attributes are most strongly correlated with memorability. Many of the features are correlated with each other (*e.g.* face visible and eye contact), suggesting a need for our feature selection strategy to have explaining-away properties.

## 4 Feature selection

Our goal is to identify a compact set of features that characterizes the memorability of an image. We note that several of our features are redundant. Some by design (such as pleasant and aesthetic) to better capture subjective notions, but others due to contextual relationships that prevail in our visual world (*e.g.* outdoor images typically contain sky). Hence, it becomes crucial that our feature selection algorithm has explaining away properties so as to determine a set of distinct characteristics that make an image memorable. Not only is this desirable via the Occam's razor view, it is also practical from an applications stand-point.

Moreover, we note that some features in our set subsume other features. For example, since the person attributes (*e.g.* hair-color) are only labeled for images containing people, they include the person presence / absence information in them. If a naive feature selection approach picked 'hair-color' as an informative feature, it would be unclear whether the mere presence or absence of a person in the image is what contributes to memorability, or if the color of the hair really matters. This issue of miscalibration of information contained in a feature also manifests itself in a more subtle manner. Our set of features include inherently multi-valued information (*e.g.* mood of the

Table 2: Attributes describing people in image

| |
|---|
| **Visibility (per-person):** Face visible? Making eye-contact? |
| **Demographics (per-person):** Gender (male, female)? Age (child, teenager, adult, senior)? Race (Caucasian, SouthEast-Asian, East-Asian, African-American, Hispanic)? |
| **Appearance (per-person):** Hair length (short, medium, long, bald)? Hair color (blonde, black, brown, red, grey)? Facial hair? |
| **Clothing (per-person):** Attire (casual, business-casual, formal)? Shirt? T-shirt? Blouse? Tie? Jacket? Sweater? Sweat-shirt? Skirt? Trousers? Shorts? A uniform? |
| **Accessories (per-person):** Dark eye-glasses? Clear eye-glasses? Hat? Earrings? Watch? Wrist jewelry? Neck jewelry? Belt? Finger Ring(s)? Make-up? |
| **Activity (per-person):** Standing? Sitting? Walking? Running? Working? Smiling? Eating? Clapping? Engaging in art? Professional activity? Buying? Selling? Giving a speech? Holding? |
| **Activity (whole image):** Sports? Adventurous? Tourist? Engaging in art? Professional? Group? |
| **Subject (whole image):** Audience? Crowd? Group? Couple? Individual? Individuals interacting? |
| **Scenario (whole image):** Routine/mundane? Unusual/strange? Pleasant? Unpleasant? Top-down? |

image), as well as inherently binary information like "a car is present in the image". It is important to calibrate the features by the amount of information captured by them.

Employing an information-theoretic approach to feature selection allows us to naturally capture both these goals: selecting a compact set of non-redundant features and calibrating features based on the information they contain.

## 4.1 Information-theoretic

We formulate our problem as that of selecting features that maximize mutual information with memorability, such that the total number of bits required to encode all selected features (i.e. the number of bits required to describe an image using the selected features) does not exceed B. Formally,

$$F^* = \arg\max \quad I\left(F; M\right)$$
$$\text{s.t.} \quad C(F) \le B \tag{1}$$

where $F$ is a subset of the features, $I\left(F; M\right)$ is the mutual information between $F$ and memorability $M$, $B$ is the budget (in bits), and $C(F)$ is the total number of bits required to encode $F$. We assume that each feature is encoded independently, and thus

$$C(F) = \sum_{i=1}^{n} C(f_i), f_i \in F \tag{2}$$

where $C(f_i)$ is the number of bits required to encode feature $f_i$, computed as $H(f_i)$, the entropy of feature $f_i$ across the training images.

This optimization is combinatorial in nature, and is NP-hard to solve. Fortunately, the work of Krause *et al.* [17] and Leskovec *et al.* [19] provides us with a computationally feasible algorithm to solve the problem. Krause *et al.* [17] showed mutual information to be a submodular function. A greedy optimization scheme to maximize submodular functions was shown to be optimal, with a constant approximation factor of $(1 - \frac{1}{e})$; i.e. no polynomial time algorithm can provide a tighter bound. Subsequently, Leskovec *et al.* [19] presented a similar greedy algorithm to select features, where each feature has a different cost associated with it (as in our set-up). The algorithm selects features with the maximum ratio of improvement in mutual information to their cost, while the total cost of the features does not exceed the allotted budget. In parallel, the cost-less version of the greedy algorithm is also used to select features (still not exceeding budget). Finally, of the two, the set of features that provides the higher mutual information is retained. This solution is at most a constant factor $\frac{1}{2}(1 - \frac{1}{e})$ away from the optimal solution [19]. Moreover, Leskovec *et al.* [19] also provided a lazy evaluation scheme that provides significant computation benefits in practice, while still maintaining the bound.

However, this lazy-greedy approach still requires the computation of mutual information between memorability and subsets of features. At each iteration, the additional information provided by a candidate feature $f_i$ over an existing set of features $F$ would be the following:

$$\text{IG}\,(f_i) = \text{I}\,(F \cup f_i; M) - \text{I}\,(F; M) \tag{3}$$

This computation is not feasible given our large number of features and limited training data. Hence, we greedily add features that maximize an approximation to the mutual information between a subset of features and memorability, as also employed by Ullman *et al.* [31]. The additional information provided by a candidate feature $f_i$ over an existing set of features $F$ is approximated as:

$$\hat{\text{IG}}\,(f_i) = \min_j \left(\text{I}\,(f_j \cup f_i; M) - \text{I}\,(f_j; M)\right), f_j \in F \tag{4}$$

The ratio of this approximation to the cost of the feature is used as the score to evaluate the usefulness of features during greedy selection. Intuitively, this ensures that the feature selected at each iteration maximizes the per-bit minimal gain in mutual information over each of the individual features already selected.

In order to maximize the mutual information (approximation) beyond the greedy algorithm, we employ multiple passes on the feature set. Given a budget $B$, we first greedily add features using a budget of $2B$, and then greedily remove features (that reduce the mutual information the least) until we fall within the allotted budget $B$. This allows for the features that were added greedily early on in the forward pass, but are explained away by subsequently added features, to be dropped. These forward and backward passes are repeated 4 times each. Note that at each pass, the objective function cannot decrease, and the final solution is still guaranteed to have a total cost within the allotted budget B.

## 4.2 Predictive

The behavior of the above approximation to mutual information has not been formally studied. While this may provide a good means to prune out many candidate features, it is unclear how close to optimal the selections will be. Feature selection within the realm of a predictive model allows us to better capture features that achieve a concrete and practical measure of performance: "which set of features allows us to make the best predictions about an image's memorability?" While selecting such features would be computationally expensive to do over all our 923 features, using a pruned set of features obtained via information-theoretic selection makes this feasible. We employ a support vector regressor (SVR, [28]) as our predictive model.

Given a set of features selected by the information-theoretic method above, we greedily select features (again, while maintaining a budget) that provide the biggest boost in regression performance (Spearman's rank correlation between predicted and ground truth memorabilities) over the training set. The same cost-based lazy-greedy selection algorithm is used as above, except with only a single pass over the feature set. This is inspired from the recent work of Das *et al.* [6], who analyzed the performance of greedy approaches to maximize submodular-like functions. They found that the submodularity ratio of a function is the best predictor of how well a greedy algorithm performs. Moreover, they found that in practice, regression performance has a high submodularity ratio, justifying the use of a greedy approach.

An alternative to greedy feature selection would be to learn a sparse-regressor. However, the parameter that controls the sparsity of the vector is not intuitive and interpretable. In the greedy feature selection approach, the budget of bits, which is interpretable, can be explicitly enforced.

## 5 Results

**Attribute annotations help:** We first tested the degree to which each general feature-type annotation in our feature set is effective at predicting memorability. We split the dataset from [13] into 2/3 training images scored by half the subjects and 1/3 test images scored by the left out half of the subjects. We trained $\epsilon$-SVRs [4] to predict memorability, using grid search to select cost and $\epsilon$ hyperparameters. For the new attributes we introduced, and for the object and scene hierarchy

features, we used RBF kernels, while for the rest of the features we used the same kernel functions as in [13]. We report performance as Spearman's rank correlation ($\rho$) between predicted and ground truth memorabilities averaged over 10 random splits of the data.

Results are shown in Table 3. We found that our new attributes annotations performed quite well ($\rho = 0.528$): they outperform higher dimensional object and scene annotations.

Table 3: Performance (rank correlation) of different types of features at predicting image memorability.

| Feature type | Perf |
|---|---|
| Object annotations | 0.494 |
| Scene annotations | 0.415 |
| Attribute annotations | 0.528 |
| Objects + Scenes + Attributes | 0.554 |

**Feature selection:** We next selected the individual best features in our set according to the feature selection algorithms described above. To compute feature entropy and mutual information, we used histogram estimators on our training data, with 7 bins per feature and 10 bins for memorability. Using these estimators, and measuring feature set cost according to (2), our entire set of 923 features has a total cost of 252 bits. We selected reduced feature sets by both running information-theoretic selection and predictive selection on our 2/3 training splits, for budgets ranging from 1 to 100-bits.

For predictive selection, we further split our training set in half and trained SVRs on one half to predict memorability on the other half. At each iteration of selection, we greedily selected the feature that maximized predictive performance averaged over 3 random splits trials, with predictive performance again measured as rank correlation between predictions and ground truth memorabilities. Since predictive selection is computational expensive, we reduced our candidate feature set by first pruning with information-theoretic selection. We took as candidates the union of all features that were selected using our information-theoretic approach for a budgets 1,2,...,100 bits. Taking this union, rather than just the features selected at a 100-bit budget, ensures that candidates were not missed when they are only effective in small budget sets.

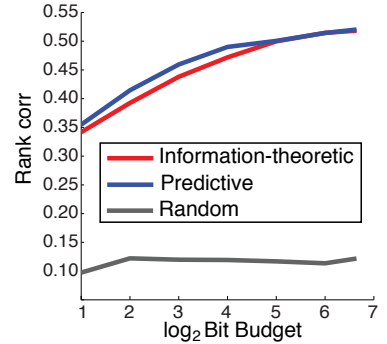

Figure 5: Regression performance vs. log bit budget of various types of feature selection. The diminishing returns (submodular-like) behavior is evident.

Next, we validated our selections on our 1/3 test set. We trained SVRs using each of our selected feature sets and made predictions on the test set. Both selection algorithms create feature sets that are similarly effective at predicting memorability (Figure 5). Using just a 16-bit budget, information-theoretic selection achieves $\rho = 0.472$, and predictive selection achieves $\rho = 0.490$ (this budget resulted in selected sets with 6 to 11 features). This performance is comparable to the performance we get using much costlier features, such as our full list of object annotations (540 features, ~106 bits, $\rho = 0.490$). As a baseline, we also compared against randomly selecting feature sets up to the same budget, which, for 16 bits, only gives $\rho = 0.119$.

We created a final list of features by running the above feature selection methods on the entire dataset (no held out data) for a budget of 10 bits. This produced the sets listed in Table 4. If one is trying to understand memorability, these features are a good place to start. In Figure 6, we explore these features further by hierarchically clustering our images according to predictive set. Each cluster can be thought of as specifying type of image with respect to memorability. For

Table 4: Information-theoretic and predictive feature selections for a budget of 10 bits. Correlations with memorability are listed after each feature (arrow indicates direction of correlation). Selections and correlations run on entire dataset.

| Information-theoretic | | Predictive | |
|---|---|---|---|
| ↑ enclosed space | 0.39 | ↑ enclosed space | 0.39 |
| ↑ face visible | 0.37 | ↑ face visible | 0.37 |
| ↓ peaceful | -0.33 | ↑ tells a story | 0.18 |
| ↓ sky present | -0.35 | ↑ recognize place | 0.16 |
| | | ↓ peaceful | -0.33 |

example, on the far right we have highly memorable "pictures of people in an enclosed space" and on the far left we have forgettable "peaceful, open, unfamiliar spaces, devoid of people."

**Automatic prediction:** While our focus in this paper is on understanding memorability, we hope that by understanding the phenomenon we may also be able to build better automatic predictors of

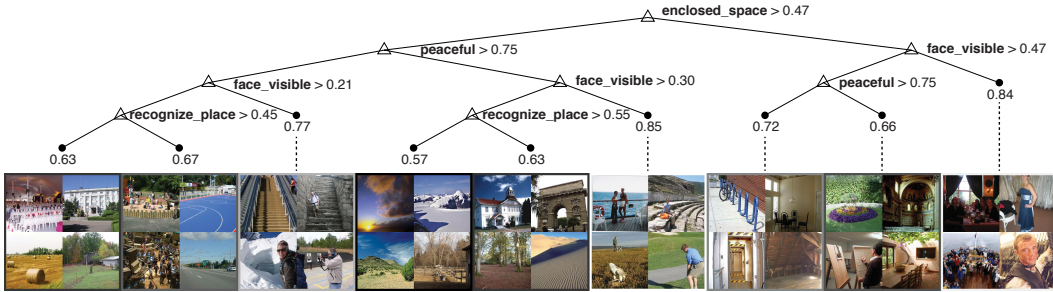

(a) Hierarchical clustering

Figure 6: Hierarchical clustering of images in 'memorability space' as achieved via a regression-tree [2], along with examples images from each cluster. Memorability of each cluster given at the leaf nodes, and also depicted as shade of cluster image borders (darker borders correspond to lower memorability than brighter borders).

it. The only previous work predicting memorability is our recent paper [13]. In that paper, we made predictions on the basis of a suite of global image features – pixel histograms, GIST, SIFT, HOG, SSIM [13]. Running the same methods on our current 2/3 data splits achieves $\rho = 0.468$. Here we attempt to do better by using our selected features as an abstraction layer between raw images and memorability.

We trained a suite of SVRs to predict annotations from images, and another SVR to predict memorability from these predicted annotations. For image features, we used the same methods as [13]. For the annotation types, we used the feature types selected by our 100-bit predictive selection on 2/3 training sets. To predict the annotations for each image in our training set, we split the training set in half and predicted annotations for one half by training on the other half, and vice versa, covering both halves with predictions. We then trained a final SVR to predict memorability on the test set

Table 5: Performance (rank correlation) of automatic memorability prediction methods.

| Features | Perf. |
|---|---|
| Direct [13] | 0.468 |
| Indirect | 0.436 |
| Direct + indirect | 0.479 |

in three ways: 1) using only image features (Direct), 2) using only predicted annotations (Indirect), and 3) using both (Direct + Indirect) (Table 5). Combining indirect predictions with direct predictions performed best ($\rho = 0.479$), slightly outperforming the direct prediction method of our previous work [13] ($\rho = 0.468$).

## 6 Conclusion

The goal of this work was to characterize aspects of an image that make it memorable. Understanding these characteristics is crucial for anyone hoping to work with memorability, be they psychologists, advertisement-designers, or photographers. We augmented the object and scene annotations of the dataset of Isola *et al.* [13] with attribute annotations describing the spatial layout, content, and aesthetic properties of the images. We employed a greedy feature selection scheme to obtain compact lists of features that are highly informative about memorability and highly predictive of memorability. We found that images of enclosed spaces containing people with visible faces are memorable, while images of vistas and peaceful settings are not. Contrary to popular belief, unusualness and aesthetic beauty attributes are not associated with high memorability – in fact, they are negatively correlated with memorability – and these attributes are not among our top few selections, indicating that other features more concisely describe memorability (Figure 4).

Through this work, we have begun to uncover some of the core features that contribute to image memorability. Understanding how these features interact to actually produce memories remains an important direction for future research. We hope that by parsing memorability into a concise and understandable set of attributes, we have provided a description that will interface well with other domains of knowledge and may provide fodder for future theories and applications of memorability.

**Acknowledgements:** We would like to thank Jianxiong Xiao for providing the global image features. This work is supported by the National Science Foundation under Grant No. 1016862 to A.O., CAREER Awards No. 0546262 to A.O and No. 0747120 to A.T. A.T. was supported in part by the Intelligence Advanced Research Projects Activity via Department of the Interior contract D10PC20023, and ONR MURI N000141010933.

## Footnotes

[1]Images (a,d,e) are among the most memorable images in our dataset, while (b,c,f) are among the least.

# References

[1] T. F. Brady, T. Konkle, G. A. Alvarez, and A. Oliva. Visual long-term memory has a massive storage capacity for object details. In *Proceedings of the National Academy of Sciences*, 2008.

[2] L. Breiman, J. Friedman, R. Olshen, and C. Stone. *Classification and regression trees*. Boca Raton, FL: CRC Press, 1984.

[3] G. D. A. Brown, I. Neath, and N. Chater. A temporal ratio model of memory. *Psych. Review*, 2007.

[4] C.-C. Chang and C.-J. Lin. *LIBSVM: a library for support vector machines*, 2001.

[5] D. Cohen-Or, O. Sorkine, R. Gal, T. Leyvand, and Y.-Q. Xu. Color harmonization. *ACM Transactions on Graphics (Proceedings of ACM SIGGRAPH)*, 2006.

[6] A. Das and D. Kempe. Submodular meets spectral: Greedy algorithms for subset selection, sparse approximation and dictionary selection. In *arXiv:1102.3975v2 [stat.ML]*, 2011.

[7] S. Dhar, V. Ordonez, and T. L. Berg. High level describable attributes for predicting aesthetics and interestingness. In *IEEE Computer Vision and Pattern Recognition*, 2011.

[8] A. Farhadi, I. Endres, D. Hoiem, and D. Forsyth. Describing objects by their attributes. In *IEEE Computer Vision and Pattern Recognition*, 2009.

[9] C. Fellbaum. Wordnet: an electronic lexical database. In *The MIT Press*, 1998.

[10] B. Gooch, E. Reinhard, C. Moulding, and P. Shirley. Artistic composition for image creation. In *Eurographics Workshop on Rendering*, 2001.

[11] M. W. Howard and M. J. Kahana. A distributed representation of temporal context. In *Journal ofMathematical Psychology*, 2001.

[12] R. R. Hunt and J. B. Worthen. Distinctiveness and memory. In *NY:Oxford Univeristy Press*, 2006.

[13] P. Isola, J. Xiao, A. Torralba, and A. Oliva. What makes an image memorable? In *IEEE Computer Vision and Pattern Recognition*, 2011.

[14] L. Itti, C. Koch, and E. Niebur. A model of saliency-based visual attention for rapid scene analysis. In *Pattern Analysis and Machine Intelligence*, 1998.

[15] T. Konkle, T. F. Brady, G. A. Alvarez, and A. Oliva. Conceptual distinctiveness supports detailed visual long-term memory for realworld objects. In *Journal of Experimental Psychology: General*, 2010.

[16] T. Konkle, T. F. Brady, G. A. Alvarez, and A. Oliva. Scene memory is more detailed than you think: the role of categories in visual longterm memory. In *Psychological Science*, 2010.

[17] A. Krause and C. Guestrin. Near-optimal nonmyopic value of information in graphical models. In *Conference on Uncertainty in Artificial Intelligence*, 2005.

[18] C. H. Lampert, H. Nickisch, and S. Harmeling. Learning to detect unseen object classes by between class attribute transfer. In *IEEE Computer Vision and Pattern Recognition*, 2009.

[19] J. Leskovec, A. Krause, C. Guestrin, C. Faloutsos, J. VanBriesen, and N. Glance. Cost-effective outbreak detection in networks. In *ACM SIGKDD International Conference on Knowledge Discovery and Data Mining*, 2007.

[20] T. Leyvand, D. Cohen-Or, G. Dror, and D. Lischinski. Data-driven enhancement of facial attractiveness. *ACM Transactions on Graphics (Proceedings of ACM SIGGRAPH 2008)*, 2008.

[21] Y. Luo and X. Tang. Photo and video quality evaluation: Focusing on the subject. In *European Conference on Computer Vision*, 2008.

[22] J. L. McClelland, B. L. McNaughton, and R. C. O'Reilly. Why there are complementary learning systems in the hippocampus and neocortex: Insights from the successes and failures of connectionist models of learning and memory. In *Psychological Review*, 1995.

[23] A. Oliva and A. Torralba. Modeling the shape of the scene: a holistic representation of the spatial envelope. In *International Journal of Computer Vision*, 2001.

[24] L. Renjie, C. L. Wolf, and D. Cohen-Or. Optimizing photo composition. In *Technical report, Tel-Aviv University*, 2010.

[25] I. Rock and P. Englestein. A study of memory for visual form. *The American Journal of Psychology*, 1959.

[26] B. C. Russell, A. Torralba, K. Murphy, and W. T. Freeman. Labelme: A database and web-based tool for image annotation. In *International Journal of Computer Vision*, 2008.

[27] R. M. Shiffrin and M. Steyvers. A model for recognition memory: Rem - retrieving effectively from memory. In *Psychnomic Bulletin and Review*, 1997.

[28] A. J. Smola and B. Schlkopf. A tutorial on support vector regression. *Statistics and Computing*, 14:199–222, 2004.

[29] M. Spain and P. Perona. Some objects are more equal than others: measuring and predicting importance. In *Proceedings of the European Conference on Computer Vision*, 2008.

[30] L. Standing. Learning 10,000 pictures. In *Quarterly Journal of Experimental Psychology*, 1973.

[31] S. Ullman, M. Vidal-Naquet, and E. Sali. Visual features of intermediate complexity and their use in classification. In *Nature Neuroscience*, 2002.

[32] J. Xiao, J. Hays, K. Ehinger, A. Oliva, and A. Torralba. Sun database: Large-scale scene recognition from abbey to zoo. In *IEEE Conference on Computer Vision and Pattern Recognition*, 2010.

